# One-unit Learning Rules for Independent Component Analysis

**Aapo Hyvärinen and Erkki Oja**
Helsinki University of Technology
Laboratory of Computer and Information Science
Rakentajanaukio 2 C, FIN-02150 Espoo, Finland
email: {Aapo.Hyvarinen,Erkki.Oja}@hut.fi

## Abstract

Neural one-unit learning rules for the problem of Independent Component Analysis (ICA) and blind source separation are introduced. In these new algorithms, every ICA neuron develops into a separator that finds one of the independent components. The learning rules use very simple constrained Hebbian/anti-Hebbian learning in which decorrelating feedback may be added. To speed up the convergence of these stochastic gradient descent rules, a novel computationally efficient fixed-point algorithm is introduced.

## 1 Introduction

Independent Component Analysis (ICA) (Comon, 1994; Jutten and Herault, 1991) is a signal processing technique whose goal is to express a set of random variables as linear combinations of statistically independent component variables. The main applications of ICA are in blind source separation, feature extraction, and blind deconvolution. In the simplest form of ICA (Comon, 1994), we observe $m$ scalar random variables $x_1, ..., x_m$ which are assumed to be linear combinations of $n$ unknown components $s_1, ... s_n$ that are zero-mean and *mutually statistically independent*. In addition, we must assume $n \leq m$. If we arrange the observed variables $x_i$ into a vector $\mathbf{x} = (x_1, x_2, ..., x_m)^T$ and the component variables $s_j$ into a vector $\mathbf{s}$, the linear relationship can be expressed as

$$\mathbf{x} = \mathbf{As} \tag{1}$$

Here, $\mathbf{A}$ is an *unknown* $m \times n$ matrix of full rank, called the mixing matrix. Noise may also be added to the model, but it is omitted here for simplicity. The basic

problem of ICA is then to estimate (separate) the realizations of the original independent components $s_j$, or a subset of them, using only the mixtures $x_i$. This is roughly equivalent to estimating the rows, or a subset of the rows, of the pseudoinverse of the mixing matrix $\mathbf{A}$. The fundamental restriction of the model is that *we can only estimate non-Gaussian independent components*, or ICs (except if just one of the ICs is Gaussian). Moreover, the ICs and the columns of $\mathbf{A}$ can only be estimated up to a multiplicative constant, because any constant multiplying an IC in eq. (1) could be cancelled by dividing the corresponding column of the mixing matrix $\mathbf{A}$ by the same constant. For mathematical convenience, we define here that the ICs $s_j$ have unit variance. This makes the (non-Gaussian) ICs unique, up to their signs. Note the assumption of zero mean of the ICs is in fact no restriction, as this can always be accomplished by subtracting the mean from the random vector $\mathbf{x}$. Note also that no order is defined between the ICs.

In *blind source separation* (Jutten and Herault, 1991), the observed values of $\mathbf{x}$ correspond to a realization of an $m$-dimensional discrete-time signal $\mathbf{x}(t)$, $t = 1, 2, ....$ Then the components $s_j(t)$ are called source signals. The source signals are usually original, uncorrupted signals or noise sources. Another application of ICA is *feature extraction* (Bell and Sejnowski, 1996; Hurri et al., 1996), where the columns of the mixing matrix $\mathbf{A}$ define features, and the $s_j$ signal the presence and the amplitude of a feature. A closely related problem is *blind deconvolution*, in which a convolved version $x(t)$ of a scalar i.i.d. signal $s(t)$ is observed. The goal is then to recover the original signal $s(t)$ without knowing the convolution kernel (Donoho, 1981). This problem can be represented in a way similar to eq. (1), replacing the matrix $\mathbf{A}$ by a filter.

The current neural algorithms for Independent Component Analysis, e.g. (Bell and Sejnowski, 1995; Cardoso and Laheld, 1996; Jutten and Herault, 1991; Karhunen et al., 1997; Oja, 1995) try to estimate simultaneously all the components. This is often not necessary, nor feasible, and it is often desired to estimate only a subset of the ICs. This is the starting point of our paper. We introduce *learning rules for a single neuron*, by which the neuron learns to estimate one of the ICs. A network of several such neurons can then estimate several (1 to $n$) ICs. Both learning rules for the 'raw' data (Section 3) and for whitened data (Section 4) are introduced. If the data is whitened, the convergence is speeded up, and some interesting simplifications and approximations are made possible. Feedback mechanisms (Section 5) are also mentioned. Finally, we introduce a novel approach for performing the computations needed in the ICA learning rules, which uses a very simple, yet highly efficient, *fixed-point iteration scheme* (Section 6). An important generalization of our learning rules is discussed in Section 7, and an illustrative experiment is shown in Section 8.

## 2 Using Kurtosis for ICA Estimation

We begin by introducing the basic mathematical framework of ICA. Most suggested solutions for ICA use the fourth-order cumulant or *kurtosis* of the signals, defined for a zero-mean random variable $v$ as $\mathrm{kurt}(v) = E\{v^4\} - 3(E\{v^2\})^2$. For a Gaussian random variable, kurtosis is zero. Therefore, random variables of positive kurtosis are sometimes called super-Gaussian, and variables of negative kurtosis sub-Gaussian. Note that for two independent random variables $v_1$ and $v_2$ and for a scalar $\alpha$, it holds $\mathrm{kurt}(v_1 + v_2) = \mathrm{kurt}(v_1) + \mathrm{kurt}(v_2)$ and $\mathrm{kurt}(\alpha v_1) = \alpha^4 \mathrm{kurt}(v_1)$.

Let us search for a linear combination of the observations $x_i$, say, $\mathbf{w}^T\mathbf{x}$, such that it has maximal or minimal kurtosis. Obviously, this is meaningful only if $\mathbf{w}$ is somehow bounded; let us assume that the variance of the linear combination is constant: $E\{(\mathbf{w}^T\mathbf{x})^2\} = 1$. Using the mixing matrix $\mathbf{A}$ in eq. (1), let us define $\mathbf{z} = \mathbf{A}^T\mathbf{w}$. Then also $\|\mathbf{z}\|^2 = \mathbf{w}^T\mathbf{A}\ \mathbf{A}^T\mathbf{w} = \mathbf{w}^T E\{\mathbf{x}\mathbf{x}^T\}\mathbf{w} = E\{(\mathbf{w}^T\mathbf{x})^2\} = 1$. Using eq. (1) and the properties of the kurtosis, we have

$$\text{kurt}(\mathbf{w}^T\mathbf{x}) = \text{kurt}(\mathbf{w}^T\mathbf{A}\mathbf{s}) = \text{kurt}(\mathbf{z}^T\mathbf{s}) = \sum_{j=1}^{n} z_j^4\,\text{kurt}(s_j) \qquad (2)$$

Under the constraint $E\{(\mathbf{w}^T\mathbf{x})^2\} = \|\mathbf{z}\|^2 = 1$, the function in (2) has a number of local minima and maxima. To make the argument clearer, let us assume for the moment that the mixture contains at least one IC whose kurtosis is negative, and at least one whose kurtosis is positive. Then, as may be obvious, and was rigorously proven by Delfosse and Loubaton (1995), the extremal points of (2) are obtained when all the components $z_j$ of $\mathbf{z}$ are zero except one component which equals $\pm 1$. In particular, the function in (2) is maximized (resp. minimized) exactly when the linear combination $\mathbf{w}^T\mathbf{x} = \mathbf{z}^T\mathbf{s}$ equals, up to the sign, one of the ICs $s_j$ of positive (resp. negative) kurtosis. Thus, *finding the extrema of kurtosis of $\mathbf{w}^T\mathbf{x}$ enables estimation of the independent components.* Equation (2) also shows that Gaussian components, or other components whose kurtosis is zero, cannot be estimated by this method.

To actually minimize or maximize $\text{kurt}(\mathbf{w}^T\mathbf{x})$, a neural algorithm based on gradient descent or ascent can be used. Then $\mathbf{w}$ is interpreted as the weight vector of a neuron with input vector $\mathbf{x}$ and linear output $\mathbf{w}^T\mathbf{x}$. The objective function can be simplified because of the constraint $E\{(\mathbf{w}^T\mathbf{x})^2\} = 1$: it holds $\text{kurt}(\mathbf{w}^T\mathbf{x}) = E\{(\mathbf{w}^T\mathbf{x})^4\} - 3$. The constraint $E\{(\mathbf{w}^T\mathbf{x})^2\} = 1$ itself can be taken into account by a penalty term. The final objective function is then of the form

$$J(\mathbf{w}) = \alpha E\{(\mathbf{w}^T\mathbf{x})^4\} + \beta F(E\{(\mathbf{w}^T\mathbf{x})^2\}) \qquad (3)$$

where $\alpha, \beta > 0$ are arbitrary scaling constants, and $F$ is a suitable penalty function. Our basic ICA learning rules are stochastic gradient descents or ascents for an objective function of this form. In the next two sections, we present learning rules resulting from adequate choices of the penalty function $F$. Preprocessing of the data (whitening) is also used to simplify $J$ in Section 4. An alternative method for finding the extrema of kurtosis is the fixed-point algorithm; see Section 6.

## 3    Basic One-Unit ICA Learning Rules

In this section, we introduce learning rules for a single neural unit. These basic learning rules require *no preprocessing of the data*, except that the data must be made zero-mean. Our learning rules are divided into two categories. As explained in Section 2, the learning rules either minimize the kurtosis of the output to separate ICs of negative kurtosis, or maximize it for components of positive kurtosis.

Let us assume that we observe a sample sequence $\mathbf{x}(t)$ of a vector $\mathbf{x}$ that is a linear combination of independent components $s_1, ..., s_n$ according to eq. (1). For separating one of the ICs of *negative kurtosis*, we use the following learning rule for

the weight vector **w** of a neuron:

$$\Delta \mathbf{w}(t) \propto \mathbf{x}(t) g^-(\mathbf{w}(t)^T \mathbf{x}(t)) \tag{4}$$

Here, the non-linear learning function $g^-$ is a simple polynomial: $g^-(u) = au - bu^3$ with arbitrary scaling constants $a, b > 0$. This learning rule is clearly a stochastic gradient descent for a function of the form (3), with $F(u) = -u$. To separate an IC of *positive kurtosis*, we use the following learning rule:

$$\Delta \mathbf{w}(t) \propto \mathbf{x}(t) g^+_{\mathbf{w}(t)}(\mathbf{w}(t)^T \mathbf{x}(t)) \tag{5}$$

where the learning function $g^+_{\mathbf{w}(t)}$ is defined as follows: $g^+_{\mathbf{w}}(u) = -au(\mathbf{w}(t)^T \mathbf{C} \mathbf{w}(t))^2 + bu^3$ where **C** is the covariance matrix of $\mathbf{x}(t)$, i.e. $\mathbf{C} = E\{\mathbf{x}(t)\mathbf{x}(t)^T\}$, and $a, b > 0$ are arbitrary constants. This learning rule is a stochastic gradient ascent for a function of the form (3), with $F(u) = -u^2$. Note that $\mathbf{w}(t)^T \mathbf{C} \mathbf{w}(t)$ in $g^+$ might also be replaced by $(E\{(\mathbf{w}(t)^T \mathbf{x}(t))^2\})^2$ or by $\|\mathbf{w}(t)\|^4$ to enable a simpler implementation.

It can be proven (Hyvärinen and Oja, 1996b) that using the learning rules (4) and (5), the linear output converges to $cs_j(t)$ where $s_j(t)$ is one of the ICs, and $c$ is a scalar constant. This multiplication of the source signal by the constant $c$ is in fact not a restriction, as the variance and the sign of the sources cannot be estimated. The only condition for convergence is that one of the ICs must be of negative (resp. positive) kurtosis, when learning rule (4) (resp. learning rule (5)) is used. Thus we can say that *the neuron learns to separate (estimate) one of the independent components*. It is also possible to combine these two learning rules into a single rule that separates an IC of any kurtosis; see (Hyvärinen and Oja, 1996b).

## 4 One-Unit ICA Learning Rules for Whitened Data

Whitening, also called sphering, is a very useful preprocessing technique. It *speeds up the convergence* considerably, makes the learning more stable numerically, and allows some interesting modifications of the learning rules. Whitening means that the observed vector **x** is linearly transformed to a vector $\mathbf{v} = \mathbf{Ux}$ such that its elements $v_i$ are mutually uncorrelated and all have unit variance (Comon, 1994). Thus the correlation matrix of **v** equals unity: $E\{\mathbf{vv}^T\} = \mathbf{I}$. This transformation is always possible and can be accomplished by classical Principal Component Analysis. At the same time, the dimensionality of the data should be reduced so that the dimension of the transformed data vector **v** equals $n$, the number of independent components. This also has the effect of reducing noise.

Let us thus suppose that the observed signal $\mathbf{v}(t)$ is whitened (sphered). Then, in order to separate one of the components of *negative kurtosis*, we can modify the learning rule (4) so as to get the following learning rule for the weight vector **w**:

$$\Delta \mathbf{w}(t) \propto \mathbf{v}(t) g^-(\mathbf{w}(t)^T \mathbf{v}(t)) - \mathbf{w}(t) \tag{6}$$

Here, the function $g^-$ is the same polynomial as above: $g^-(u) = au - bu^3$ with $a > 1$ and $b > 0$. This modification is valid because we now have $E\mathbf{v}(\mathbf{w}^T\mathbf{v}) = \mathbf{w}$ and thus we can add $+\mathbf{w}(t)$ in the linear part of $g^-$ and subtract $\mathbf{w}(t)$ explicitly afterwards. The modification is useful because it allows us to approximate $g^-$ with

the 'tanh' function, as $\mathbf{w}(t)^T\mathbf{v}(t)$ then stays in the range where this approximation is valid. Thus we get what is perhaps the simplest possible stable Hebbian learning rule for a nonlinear Perceptron.

To separate one of the components of *positive kurtosis*, rule (5) simplifies to:

$$\Delta\mathbf{w}(t) \propto b\mathbf{v}(t)(\mathbf{w}(t)^T\mathbf{v}(t))^3 - a\|\mathbf{w}(t)\|^4\mathbf{w}(t). \tag{7}$$

## 5 Multi-Unit ICA Learning Rules

If estimation of several independent components is desired, it is possible to construct a neural network by combining $N$ $(1 \le N \le n)$ neurons that learn according to the learning rules given above, and *adding a feedback* term to each of those learning rules. A discussion of such networks can be found in (Hyvärinen and Oja, 1996b).

## 6 Fixed-Point Algorithm for ICA

The advantage of neural on-line learning rules like those introduced above is that the inputs $\mathbf{v}(t)$ can be used in the algorithm at once, thus enabling faster adaptation in a non-stationary environment. A resulting trade-off, however, is that the convergence is slow, and depends on a good choice of the learning rate sequence, i.e. the step size at each iteration. A bad choice of the learning rate can, in practice, destroy convergence. Therefore, some ways to *make the learning radically faster and more reliable* may be needed. The fixed-point iteration algorithms are such an alternative. Based on the learning rules introduced above, we introduce here a fixed-point algorithm, whose convergence is proven and analyzed in detail in (Hyvärinen and Oja, 1997). For simplicity, we only consider the case of whitened data here.

Consider the general neural learning rule trying to find the extrema of kurtosis. In a fixed point of such a learning rule, the sum of the gradient of kurtosis and the penalty term must equal zero: $E\{\mathbf{v}(\mathbf{w}^T\mathbf{v})^3\} - 3\|\mathbf{w}\|^2\mathbf{w} + f(\|\mathbf{w}\|^2)\mathbf{w} = 0$. The solutions of this equation must satisfy

$$\mathbf{w} = \text{scalar} \times (E\{\mathbf{v}(\mathbf{w}^T\mathbf{v})^3\} - 3\mathbf{w}\|\mathbf{w}\|^2) \tag{8}$$

Actually, because the norm of $\mathbf{w}$ is irrelevant, it is the direction of the right hand side that is important. Therefore the scalar in eq. (8) is not significant and its effect can be replaced by explicit normalization.

Assume now that we have collected a sample of the random vector $\mathbf{v}$, which is a whitened (or sphered) version of the vector $\mathbf{x}$ in eq. (1). Using (8), we obtain the following *fixed-point algorithm for ICA*:

  1. Take a random initial vector $\mathbf{w}(0)$ of norm 1. Let $k = 1$.

  2. Let $\mathbf{w}(k) = E\{\mathbf{v}(\mathbf{w}(k-1)^T\mathbf{v})^3\} - 3\mathbf{w}(k-1)$. The expectation can be estimated using a large sample of $\mathbf{v}$ vectors (say, 1,000 points).

  3. Divide $\mathbf{w}(k)$ by its norm.

  4. If $|\mathbf{w}(k)^T\mathbf{w}(k-1)|$ is not close enough to 1, let $k = k+1$ and go back to step 2. Otherwise, output the vector $\mathbf{w}(k)$.

The final vector $\mathbf{w}^* = \lim_k \mathbf{w}(k)$ given by the algorithm separates one of the non-Gaussian ICs in the sense that $\mathbf{w}^{*T}\mathbf{v}$ equals one of the ICs $s_j$. No distinction between components of positive or negative kurtosis is needed here. A remarkable property of our algorithm is that a very small number of iterations, usually 5-10, seems to be enough to obtain the maximal accuracy allowed by the sample data. This is due to the fact that the convergence of the fixed point algorithm is in fact *cubic*, as shown in (Hyvärinen and Oja, 1997).

To estimate $N$ ICs, we run this algorithm $N$ times. To ensure that we estimate each time a different IC, we only need to add a simple projection inside the loop, which forces the solution vector $\mathbf{w}(k)$ to be orthogonal to the previously found solutions. This is possible because the desired weight vectors are orthonormal for whitened data (Hyvärinen and Oja, 1996b; Karhunen et al., 1997). Symmetric methods of orthogonalization may also be used (Hyvärinen, 1997).

This fixed-point algorithm has several advantages when compared to other suggested ICA methods. First, the convergence of our algorithm is cubic. This means very fast convergence and is rather unique among the ICA algorithms. Second, contrary to gradient-based algorithms, there is no learning rate or other adjustable parameters in the algorithm, which makes it easy to use and more reliable. Third, components of both positive and negative kurtosis can be directly estimated by the same fixed-point algorithm.

# 7 Generalizations of Kurtosis

In the learning rules introduced above, we used kurtosis as an optimization criterion for ICA estimation. This approach can be generalized to a large class of such optimizaton criteria, called contrast functions. For the case of on-line learning rules, this approach is developed in (Hyvärinen and Oja, 1996a), in which it is shown that the function $g$ in the learning rules in section 4 can be, in fact, replaced by practically any non-linear function (provided that $\mathbf{w}$ is normalized properly). Whether one must use Hebbian or anti-Hebbian learning is then determined by the sign of certain 'non-polynomial cumulants'. The utility of such a generalization is that one can then choose the non-linearity according to some statistical optimality criteria, such as *robustness against outliers*.

The fixed-point algorithm may also be generalized for an arbitrary non-linearity, say $g$. Step 2 in the fixed-point algorithm then becomes (for whitened data) (Hyvärinen, 1997): $\mathbf{w}(k) = E\{\mathbf{v}g(\mathbf{w}(k-1)^T\mathbf{v})\} - E\{g'(\mathbf{w}(k-1)^T\mathbf{v})\}\mathbf{w}(k-1)$.

# 8 Experiments

A visually appealing way of demonstrating how ICA algorithms work is to use them to separate images from their linear mixtures. On the left in Fig. 1, four superimposed mixtures of 4 unknown images are depicted. Defining the $j$-th IC $s_j$ to be the gray-level value of the $j$-th image in a given position, and scanning the 4 images simultaneously pixel by pixel, we can use the ICA model and recover the original images. For example, we ran the fixed-point algorithm four times, estimating the four images shown on the right in Fig. 1. The algorithm needed on the average 7 iterations for each IC.

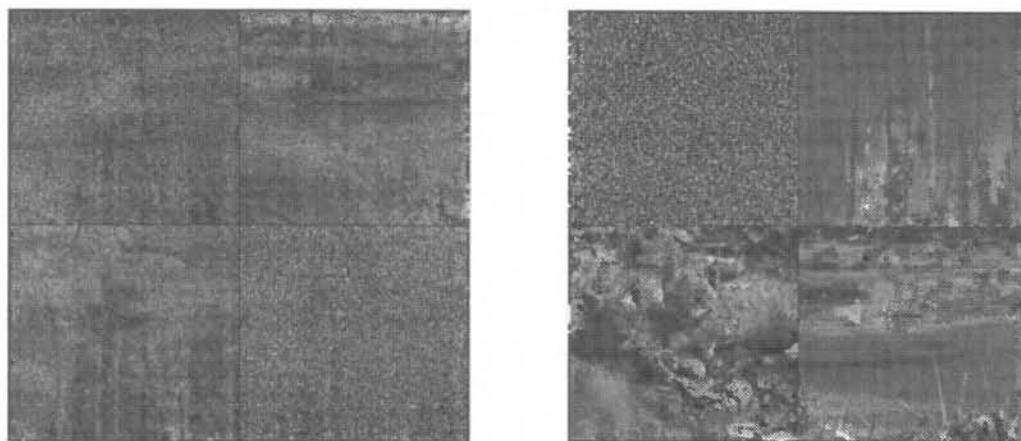

Figure 1: Three photographs of natural scenes and a noise image were linearly mixed to illustrate our algorithms. The mixtures are depicted on the left. On the right, the images recovered by the fixed-point algorithm are shown.

# References

Bell, A. and Sejnowski, T. (1995). An information-maximization approach to blind separation and blind deconvolution. *Neural Computation*, 7:1129–1159.

Bell, A. and Sejnowski, T. J. (1996). Edges are the independent components of natural scenes. In *NIPS*96*, Denver, Colorado.

Cardoso, J.-F. and Laheld, B. H. (1996). Equivariant adaptive source separation. *IEEE Trans. on Signal Processing*, 44(12).

Comon, P. (1994). Independent component analysis – a new concept? *Signal Processing*, 36:287–314.

Delfosse, N. and Loubaton, P. (1995). Adaptive blind separation of independent sources: a deflation approach. *Signal Processing*, 45:59–83.

Donoho, D. (1981). On minimum entropy deconvolution. In *Applied Time Series Analysis II*. Academic Press.

Hurri, J., Hyvärinen, A., Karhunen, J., and Oja, E. (1996). Image feature extraction using independent component analysis. In *Proc. NORSIG'96*, Espoo, Finland.

Hyvärinen, A. (1997). A family of fixed-point algorithms for independent component analysis. In *Proc. ICASSP'97*, Munich, Germany.

Hyvärinen, A. and Oja, E. (1996a). Independent component analysis by general nonlinear hebbian-like learning rules. Technical Report A41, Helsinki University of Technology, Laboratory of Computer and Information Science.

Hyvärinen, A. and Oja, E. (1996b). Simple neuron models for independent component analysis. Technical Report A37, Helsinki University of Technology, Laboratory of Computer and Information Science.

Hyvärinen, A. and Oja, E. (1997). A fast fixed-point algorithm for independent component analysis. *Neural Computation*. To appear.

Jutten, C. and Herault, J. (1991). Blind separation of sources, part I: An adaptive algorithm based on neuromimetic architecture. *Signal Processing*, 24:1–10.

Karhunen, J., Oja, E., Wang, L., Vigario, R., and Joutsensalo, J. (1997). A class of neural networks for independent component analysis. *IEEE Trans. on Neural Networks*. To appear.

Oja, E. (1995). The nonlinear PCA learning rule and signal separation – mathematical analysis. Technical Report A 26, Helsinki University of Technology, Laboratory of Computer and Information Science. Submitted to a journal.
